# TD(0) Leads to Better Policies than Approximate Value Iteration

**Benjamin Van Roy**

Management Science and Engineering and Electrical Engineering

Stanford University

Stanford, CA 94305

`bvr@stanford.edu`

## Abstract

We consider approximate value iteration with a parameterized approximator in which the state space is partitioned and the optimal cost-to-go function over each partition is approximated by a constant. We establish performance loss bounds for policies derived from approximations associated with fixed points. These bounds identify benefits to having projection weights equal to the invariant distribution of the resulting policy. Such projection weighting leads to the same fixed points as TD(0). Our analysis also leads to the first performance loss bound for approximate value iteration with an average cost objective.

## 1 Preliminaries

Consider a discrete-time communicating Markov decision process (MDP) with a finite state space $\mathcal{S} = \{1, \ldots, |\mathcal{S}|\}$. At each state $x \in \mathcal{S}$, there is a finite set $\mathcal{U}_x$ of admissible actions. If the current state is $x$ and an action $u \in \mathcal{U}_x$ is selected, a cost of $g_u(x)$ is incurred, and the system transitions to a state $y \in \mathcal{S}$ with probability $p_{xy}(u)$. For any $x \in \mathcal{S}$ and $u \in \mathcal{U}_x$, $\sum_{y \in \mathcal{S}} p_{xy}(u) = 1$. Costs are discounted at a rate of $\alpha \in (0,1)$ per period. Each instance of such an MDP is defined by a quintuple $(\mathcal{S}, \mathcal{U}, g, p, \alpha)$.

A (stationary deterministic) policy is a mapping $\mu$ that assigns an action $u \in \mathcal{U}_x$ to each state $x \in \mathcal{S}$. If actions are selected based on a policy $\mu$, the state follows a Markov process with transition matrix $P_\mu$, where each $(x,y)$th entry is equal to $p_{xy}(\mu(x))$. The restriction to communicating MDPs ensures that it is possible to reach any state from any other state.

Each policy $\mu$ is associated with a cost-to-go function $J_\mu \in \Re^{|\mathcal{S}|}$, defined by $J_\mu = \sum_{t=0}^\infty \alpha^t P_\mu^t g_\mu = (I - \alpha P_\mu)^{-1} g_\mu$, where, with some abuse of notation, $g_\mu(x) = g_{\mu(x)}(x)$ for each $x \in \mathcal{S}$. A policy $\mu$ is said to be *greedy* with respect to a function $J$ if $\mu(x) \in \operatorname*{argmin}_{u \in \mathcal{U}_x}(g_u(x) + \alpha \sum_{y \in \mathcal{S}} p_{xy}(u)J(y))$ for all $x \in \mathcal{S}$.

The optimal cost-to-go function $J^* \in \Re^{|\mathcal{S}|}$ is defined by $J^*(x) = \min_\mu J_\mu(x)$, for all $x \in \mathcal{S}$. A policy $\mu^*$ is said to be optimal if $J_{\mu^*} = J^*$. It is well-known that an optimal policy exists. Further, a policy $\mu^*$ is optimal if and only if it is greedy with respect to $J^*$. Hence, given the optimal cost-to-go function, optimal actions can computed be minimizing the right-hand side of the above inclusion.

Value iteration generates a sequence $J_\ell$ converging to $J^*$ according to $J_{\ell+1} = TJ_\ell$, where $T$ is the dynamic programming operator, defined by $(TJ)(x) = \min_{u \in \mathcal{U}_x}(g_u(x) + \alpha \sum_{y \in \mathcal{S}} p_{xy}(u)J(y))$, for all $x \in \mathcal{S}$ and $J \in \Re^{|\mathcal{S}|}$. This sequence converges to $J^*$ for any initialization of $J_0$.

## 2  Approximate Value Iteration

The state spaces of relevant MDPs are typically so large that computation and storage of a cost-to-go function is infeasible. One approach to dealing with this obstacle involves partitioning the state space $\mathcal{S}$ into a manageable number $K$ of disjoint subsets $\mathcal{S}_1, \ldots, \mathcal{S}_K$ and approximating the optimal cost-to-go function with a function that is constant over each partition. This can be thought of as a form of state aggregation – all states within a given partition are assumed to share a common optimal cost-to-go.

To represent an approximation, we define a matrix $\Phi \in \Re^{|\mathcal{S}| \times K}$ such that each $k$th column is an indicator function for the $k$th partition $\mathcal{S}_k$. Hence, for any $r \in \Re^K$, $k$, and $x \in \mathcal{S}_k$, $(\Phi r)(x) = r_k$. In this paper, we study variations of value iteration, each of which computes a vector $r$ so that $\Phi r$ approximates $J^*$. The use of such a policy $\mu_r$ which is greedy with respect to $\Phi r$ is justified by the following result (see [10] for a proof):

**Theorem 1** *If $\mu$ is a greedy policy with respect to a function $\tilde{J} \in \Re^{|\mathcal{S}|}$ then*

$$\|J_\mu - J^*\|_\infty \leq \frac{2\alpha}{1-\alpha}\|J^* - \tilde{J}\|_\infty.$$

One common way of approximating a function $J \in \Re^{|\mathcal{S}|}$ with a function of the form $\Phi r$ involves projection with respect to a weighted Euclidean norm $\|\cdot\|_\pi$. The weighted Euclidean norm: $\|J\|_{2,\pi} = \left(\sum_{x \in \mathcal{S}} \pi(x)J^2(x)\right)^{1/2}$. Here, $\pi \in \Re_+^{|\mathcal{S}|}$ is a vector of weights that assign relative emphasis among states. The projection $\Pi_\pi J$ is the function $\Phi r$ that attains the minimum of $\|J - \Phi r\|_{2,\pi}$; if there are multiple functions $\Phi r$ that attain the minimum, they must form an affine space, and the projection is taken to be the one with minimal norm $\|\Phi r\|_{2,\pi}$. Note that in our context, where each $k$th column of $\Phi$ represents an indicator function for the $k$th partition, for any $\pi$, $J$, and $x \in \mathcal{S}_k$, $(\Pi_\pi J)(x) = \sum_{y \in \mathcal{S}_k} \pi(y)J(y) / \sum_{y \in \mathcal{S}_k} \pi(y)$.

Approximate value iteration begins with a function $\Phi r^{(0)}$ and generates a sequence according to $\Phi r^{(\ell+1)} = \Pi_\pi T \Phi r^{(\ell)}$. It is well-known that the dynamic programming operator $T$ is a contraction mapping with respect to the maximum norm. Further, $\Pi_\pi$ is maximum-norm nonexpansive [16, 7, 8]. (This is not true for general $\Phi$, but is true in our context in which columns of $\Phi$ are indicator functions for partitions.) It follows that the composition $\Pi_\pi T$ is a contraction mapping. By the contraction mapping theorem, $\Pi_\pi T$ has a unique fixed point $\Phi \tilde{r}$, which is the limit of the sequence $\Phi r^{(\ell)}$. Further, the following result holds:

**Theorem 2** *For any MDP, partition, and weights $\pi$ with support intersecting every partition, if $\Phi \tilde{r} = \Pi_\pi T \Phi \tilde{r}$ then*

$$\|\Phi \tilde{r} - J^*\|_\infty \leq \frac{2}{1-\alpha} \min_{r \in \Re^K} \|J^* - \Phi r\|_\infty,$$

*and*

$$(1-\alpha)\|J_{\mu_{\tilde{r}}} - J^*\|_\infty \leq \frac{4\alpha}{1-\alpha} \min_{r \in \Re^K} \|J^* - \Phi r\|_\infty.$$

The first inequality of the theorem is an *approximation error bound*, established in [16, 7, 8] for broader classes of approximators that include state aggregation as a special case. The

second is a *performance loss bound*, derived by simply combining the approximation error bound and Theorem 1.

Note that $J_{\mu_{\tilde{r}}}(x) \geq J^*(x)$ for all $x$, so the left-hand side of the performance loss bound is the maximal increase in cost-to-go, normalized by $1 - \alpha$. This normalization is natural, since a cost-to-go function is a linear combination of expected future costs, with coefficients $1, \alpha, \alpha^2, \ldots$, which sum to $1/(1 - \alpha)$.

Our motivation of the normalizing constant begs the question of whether, for fixed MDP parameters $(\mathcal{S}, \mathcal{U}, g, p)$ and fixed $\Phi$, $\min_r \|J^* - \Phi r\|_\infty$ also grows with $1/(1 - \alpha)$. It turns out that $\min_r \|J^* - \Phi r\|_\infty = O(1)$. To see why, note that for any $\mu$,

$$J_\mu = (I - \alpha P_\mu)^{-1} g_\mu = \frac{1}{1 - \alpha} \lambda_\mu + h_\mu,$$

where $\lambda_\mu(x)$ is the *expected average cost* if the process starts in state $x$ and is controlled by policy $\mu$,

$$\lambda_\mu = \lim_{\tau \to \infty} \frac{1}{\tau} \sum_{t=0}^{\tau-1} P_\mu^t g_\mu,$$

and $h_\mu$ is the *discounted differential cost function*

$$h_\mu = (I - \alpha P_\mu)^{-1} (g_\mu - \lambda_\mu).$$

Both $\lambda_\mu$ and $h_\mu$ converge to finite vectors as $\alpha$ approaches 1 [3]. For an optimal policy $\mu^*$, $\lim_{\alpha \uparrow 1} \lambda_{\mu^*}(x)$ does not depend on $x$ (in our context of a communicating MDP). Since constant functions lie in the range of $\Phi$,

$$\lim_{\alpha \uparrow 1} \min_{r \in \Re^K} \|J^* - \Phi r\|_\infty \leq \lim_{\alpha \uparrow 1} \|h_{\mu^*}\|_\infty < \infty.$$

The performance loss bound still exhibits an undesirable dependence on $\alpha$ through the coefficient $4\alpha/(1 - \alpha)$. In most relevant contexts, $\alpha$ is close to 1; a representative value might be 0.99. Consequently, $4\alpha/(1 - \alpha)$ can be very large. Unfortunately, the bound is sharp, as expressed by the following theorem. We will denote by $\mathbf{1}$ the vector with every component equal to 1.

**Theorem 3** *For any $\delta > 0$, $\alpha \in (0, 1)$, and $\Delta \geq 0$, there exists MDP parameters $(\mathcal{S}, \mathcal{U}, g, p)$ and a partition such that $\min_{r \in \Re^K} \|J^* - \Phi r\|_\infty = \Delta$ and, if $\Phi \tilde{r} = \Pi_\pi T \Phi \tilde{r}$ with $\pi = \mathbf{1}$,*

$$(1 - \alpha) \|J_{\mu_{\tilde{r}}} - J^*\|_\infty \geq \frac{4\alpha}{1 - \alpha} \min_{r \in \Re^K} \|J^* - \Phi r\|_\infty - \delta.$$

This theorem is established through an example in [22]. The choice of uniform weights ($\pi = \mathbf{1}$) is meant to point out that even for such a simple, perhaps natural, choice of weights, the performance loss bound is sharp.

Based on Theorems 2 and 3, one might expect that there exists MDP parameters $(\mathcal{S}, \mathcal{U}, g, p)$ and a partition such that, with $\pi = \mathbf{1}$,

$$(1 - \alpha) \|J_{\mu_{\tilde{r}}} - J^*\|_\infty = \Theta \left( \frac{1}{1 - \alpha} \min_{r \in \Re^K} \|J^* - \Phi r\|_\infty \right).$$

In other words, that the performance loss is both lower and upper bounded by $1/(1 - \alpha)$ times the smallest possible approximation error. It turns out that this is not true, at least if we restrict to a finite state space. However, as the following theorem establishes, the coefficient multiplying $\min_{r \in \Re^K} \|J^* - \Phi r\|_\infty$ can grow arbitrarily large as $\alpha$ increases, keeping all else fixed.

**Theorem 4** *For any $L$ and $\Delta \geq 0$, there exists MDP parameters $(\mathcal{S}, \mathcal{U}, g, p)$ and a partition such that $\lim_{\alpha \uparrow 1} \min_{r \in \Re^K} \|J^* - \Phi r\|_\infty = \Delta$ and, if $\Phi \tilde{r} = \Pi_\pi T \Phi \tilde{r}$ with $\pi = \mathbf{1}$,*

$$\liminf_{\alpha \uparrow 1}(1 - \alpha)\left(J_{\mu_{\tilde{r}}}(x) - J^*(x)\right) \geq L \lim_{\alpha \uparrow 1} \min_{r \in \Re^K} \|J^* - \Phi r\|_\infty,$$

*for all $x \in \mathcal{S}$.*

This Theorem is also established through an example [22].

For any $\mu$ and $x$,

$$\lim_{\alpha \uparrow 1}\left((1 - \alpha)J_\mu(x) - \lambda_\mu(x)\right) = \lim_{\alpha \uparrow 1}(1 - \alpha)h_\mu(x) = 0.$$

Combined with Theorem 4, this yields the following corollary.

**Corollary 1** *For any $L$ and $\Delta \geq 0$, there exists MDP parameters $(\mathcal{S}, \mathcal{U}, g, p)$ and a partition such that $\lim_{\alpha \uparrow 1} \min_{r \in \Re^K} \|J^* - \Phi r\|_\infty = \Delta$ and, if $\Phi \tilde{r} = \Pi_\pi T \Phi \tilde{r}$ with $\pi = \mathbf{1}$,*

$$\liminf_{\alpha \uparrow 1}\left(\lambda_{\mu_{\tilde{r}}}(x) - \lambda_{\mu^*}(x)\right) \geq L \lim_{\alpha \uparrow 1} \min_{r \in \Re^K} \|J^* - \Phi r\|_\infty,$$

*for all $x \in \mathcal{S}$.*

## 3 Using the Invariant Distribution

In the previous section, we considered an approximation $\Phi \tilde{r}$ that solves $\Pi_\pi T \Phi \tilde{r} = \Phi \tilde{r}$ for some arbitrary pre-selected weights $\pi$. We now turn to consider use of an invariant state distribution $\pi_{\tilde{r}}$ of $P_{\mu_{\tilde{r}}}$ as the weight vector.[1] This leads to a circular definition: the weights are used in defining $\tilde{r}$ and now we are defining the weights in terms of $\tilde{r}$. What we are really after here is a vector $\tilde{r}$ that satisfies $\Pi_{\pi_{\tilde{r}}} T \Phi \tilde{r} = \Phi \tilde{r}$. The following theorem captures the associated benefits. (Due to space limitations, we omit the proof, which is provided in the full length version of this paper [22].)

**Theorem 5** *For any MDP and partition, if $\Phi \tilde{r} = \Pi_{\pi_{\tilde{r}}} T \Phi \tilde{r}$ and $\pi_{\tilde{r}}$ has support intersecting every partition, $(1 - \alpha)\pi_{\tilde{r}}^T (J_{\mu_{\tilde{r}}} - J^*) \leq 2\alpha \min_{r \in \Re^K} \|J^* - \Phi r\|_\infty$.*

When $\alpha$ is close to 1, which is typical, the right-hand side of our new performance loss bound is far less than that of Theorem 2. The primary improvement is in the omission of a factor of $1 - \alpha$ from the denominator. But for the bounds to be compared in a meaningful way, we must also relate the left-hand-side expressions. A relation can be based on the fact that for all $\mu$, $\lim_{\alpha \uparrow 1} \|(1 - \alpha)J_\mu - \lambda_\mu\|_\infty = 0$, as explained in Section 2. In particular, based on this, we have

$$\lim_{\alpha \uparrow 1}(1 - \alpha)\|J_\mu - J^*\|_\infty = |\lambda_\mu - \lambda^*| = \lambda_\mu - \lambda^* = \lim_{\alpha \uparrow 1} \pi^T (J_\mu - J^*),$$

for all policies $\mu$ and probability distributions $\pi$. Hence, the left-hand-side expressions from the two performance bounds become directly comparable as $\alpha$ approaches 1.

Another interesting comparison can be made by contrasting Corollary 1 against the following immediate consequence of Theorem 5.

**Corollary 2** *For all MDP parameters $(\mathcal{S}, \mathcal{U}, g, p)$ and partitions, if $\Phi \tilde{r} = \Pi_{\pi_{\tilde{r}}} T \Phi \tilde{r}$ and $\liminf_{\alpha \uparrow 1} \sum_{x \in \mathcal{S}_k} \pi_{\tilde{r}}(x) > 0$ for all $k$,*

$$\limsup_{\alpha \uparrow 1} \|\lambda_{\mu_{\tilde{r}}} - \lambda_{\mu^*}\|_\infty \leq 2 \lim_{\alpha \uparrow 1} \min_{r \in \Re^K} \|J^* - \Phi r\|_\infty.$$

The comparison suggests that solving $\Phi \tilde{r} = \Pi_{\pi_{\tilde{r}}} T \Phi \tilde{r}$ is strongly preferable to solving $\Phi \tilde{r} = \Pi_\pi T \Phi \tilde{r}$ with $\pi = \mathbf{1}$.

## 4  Exploration

If a vector $\tilde{r}$ solves $\Phi\tilde{r} = \Pi_{\pi_{\tilde{r}}}T\Phi\tilde{r}$ and the support of $\pi_{\tilde{r}}$ intersects every partition, Theorem 5 promises a desirable bound. However, there are two significant shortcomings to this solution concept, which we will address in this section. First, in some cases, the equation $\Pi_{\pi_{\tilde{r}}}T\Phi\tilde{r} = \Phi\tilde{r}$ does not have a solution. It is easy to produce examples of this; though no example has been documented for the particular class of approximators we are using here, [2] offers an example involving a different linearly parameterized approximator that captures the spirit of what can happen. Second, it would be nice to relax the requirement that the support of $\pi_{\tilde{r}}$ intersect every partition.

To address these shortcomings, we introduce stochastic policies. A stochastic policy $\mu$ maps state-action pairs to probabilities. For each $x \in \mathcal{S}$ and $u \in \mathcal{U}_x$, $\mu(x, u)$ is the probability of taking action $u$ when in state $x$. Hence, $\mu(x, u) \geq 0$ for all $x \in \mathcal{S}$ and $u \in \mathcal{U}_x$, and $\sum_{u \in \mathcal{U}_x} \mu(x, u) = 1$ for all $x \in \mathcal{S}$.

Given a scalar $\epsilon > 0$ and a function $J$, the $\epsilon$-greedy Boltzmann exploration policy with respect to $J$ is defined by

$$\mu(x, u) = \frac{e^{-(T_u J)(x)(|\mathcal{U}_x|-1)/\epsilon e}}{\sum_{u \in \mathcal{U}_x} e^{-(T_u J)(x)(|\mathcal{U}_x|-1)/\epsilon e}}.$$

For any $\epsilon > 0$ and $r$, let $\mu_r^{\epsilon}$ denote the $\epsilon$-greedy Boltzmann exploration policy with respect to $\Phi r$. Further, we define a modified dynamic programming operator that incorporates Boltzmann exploration:

$$(T^{\epsilon}J)(x) = \frac{\sum_{u \in \mathcal{U}_x} e^{-(T_u J)(x)(|\mathcal{U}_x|-1)/\epsilon e}(T_u J)(x)}{\sum_{u \in \mathcal{U}_x} e^{-(T_u J)(x)(|\mathcal{U}_x|-1)/\epsilon e}}.$$

As $\epsilon$ approaches 0, $\epsilon$-greedy Boltzmann exploration policies become greedy and the modified dynamic programming operators become the dynamic programming operator. More precisely, for all $r$, $x$, and $J$, $\lim_{\epsilon\downarrow 0} \mu_r^{\epsilon}(x, \mu_r(x)) = 1$ and $\lim_{\epsilon\downarrow 1} T^{\epsilon}J = TJ$. These are immediate consequences of the following result (see [4] for a proof).

**Lemma 1** *For any $n$, $v \in \Re^n$, $\min_i v_i + \epsilon \geq \sum_i e^{-v_i(n-1)/\epsilon e}v_i / \sum_i e^{-v_i(n-1)/\epsilon e} \geq \min_i v_i$.*

Because we are only concerned with communicating MDPs, there is a unique invariant state distribution associated with each $\epsilon$-greedy Boltzmann exploration policy $\mu_r^{\epsilon}$ and the support of this distribution is $\mathcal{S}$. Let $\pi_r^{\epsilon}$ denote this distribution. We consider a vector $\tilde{r}$ that solves $\Phi\tilde{r} = \Pi_{\pi_{\tilde{r}}^{\epsilon}}T^{\epsilon}\Phi\tilde{r}$. For any $\epsilon > 0$, there exists a solution to this equation (this is an immediate extension of Theorem 5.1 from [4]).

We have the following performance loss bound, which parallels Theorem 5 but with an equation for which a solution is guaranteed to exist and without any requirement on the resulting invariant distribution. (Again, we omit the proof, which is available in [22].)

**Theorem 6** *For any MDP, partition, and $\epsilon > 0$, if $\Phi\tilde{r} = \Pi_{\pi_{\tilde{r}}^{\epsilon}}T^{\epsilon}\Phi\tilde{r}$ then $(1 - \alpha)(\pi_{\tilde{r}}^{\epsilon})^T(J_{\mu_{\tilde{r}}^{\epsilon}} - J^*) \leq 2\alpha \min_{r \in \Re^K} \|J^* - \Phi r\|_{\infty} + \epsilon$.*

## 5  Computation: TD(0)

Though computation is not a focus of this paper, we offer a brief discussion here. First, we describe a simple algorithm from [16], which draws on ideas from temporal-difference learning [11, 12] and Q-learning [23, 24] to solve $\Phi\tilde{r} = \Pi_{\pi}T\Phi\tilde{r}$. It requires an ability to sample a sequence of states $x^{(0)}, x^{(1)}, x^{(2)}, \ldots$, each independent and identically

distributed according to $\pi$. Also required is a way to efficiently compute $(T\Phi r)(x) = \min_{u \in \mathcal{U}_x}(g_u(x) + \alpha \sum_{y \in \mathcal{S}} p_{xy}(u)(\Phi r)(y))$, for any given $x$ and $r$. This is typically possible when the action set $\mathcal{U}_x$ and the support of $p_{x\cdot}(u)$ (i.e., the set of states that can follow $x$ if action $u$ is selected) are not too large. The algorithm generates a sequence of vectors $r^{(\ell)}$ according to

$$ r^{(\ell+1)} = r^{(\ell)} + \gamma_\ell \phi(x^{(\ell)})\left((T\Phi r^{(\ell)})(x^{(\ell)}) - (\Phi r^{(\ell)})(x^{(\ell)})\right), $$

where $\gamma_\ell$ is a step size and $\phi(x)$ denotes the column vector made up of components from the $x$th row of $\Phi$. In [16], using results from [15, 9], it is shown that under appropriate assumptions on the step size sequence, $r^{(\ell)}$ converges to a vector $\tilde{r}$ that solves $\Phi\tilde{r} = \Pi_\pi T\Phi\tilde{r}$.

The equation $\Phi\tilde{r} = \Pi_\pi T\Phi\tilde{r}$ may have no solution. Further, the requirement that states are sampled independently from the invariant distribution may be impractical. However, a natural extension of the above algorithm leads to an easily implementable version of TD(0) that aims at solving $\Phi\tilde{r} = \Pi_{\pi_{\tilde{r}}^\epsilon}T^\epsilon\Phi\tilde{r}$. The algorithm requires simulation of a trajectory $x_0, x_1, x_2, \ldots$ of the MDP, with each action $u_t \in \mathcal{U}_{x_t}$ generated by the $\epsilon$-greedy Boltzmann exploration policy with respect to $\Phi r^{(t)}$. The sequence of vectors $r^{(t)}$ is generated according to

$$ r^{(t+1)} = r^{(t)} + \gamma_t \phi(x_t)\left((T^\epsilon \Phi r^{(t)})(x_t) - (\Phi r^{(t)})(x_t)\right). $$

Under suitable conditions on the step size sequence, if this algorithm converges, the limit satisfies $\Phi\tilde{r} = \Pi_{\pi_{\tilde{r}}^\epsilon}T^\epsilon\Phi\tilde{r}$. Whether such an algorithm converges and whether there are other algorithms that can effectively solve $\Phi\tilde{r} = \Pi_{\pi_{\tilde{r}}^\epsilon}T^\epsilon\Phi\tilde{r}$ for broad classes of relevant problems remain open issues.

## 6  Extensions and Open Issues

Our results demonstrate that weighting a Euclidean norm projection by the invariant distribution of a greedy (or approximately greedy) policy can lead to a dramatic performance gain. It is intriguing that temporal-difference learning implicitly carries out such a projection, and consequently, any limit of convergence obeys the stronger performance loss bound.

This is not the first time that the invariant distribution has been shown to play a critical role in approximate value iteration and temporal-difference learning. In prior work involving approximation of a cost-to-go function for a fixed policy (no control) and a general linearly parameterized approximator (arbitrary matrix $\Phi$), it was shown that weighting by the invariant distribution is key to ensuring convergence and an approximation error bound [17, 18]. Earlier empirical work anticipated this [13, 14].

The temporal-difference learning algorithm presented in Section 5 is a version of TD(0), This is a special case of TD($\lambda$), which is parameterized by $\lambda \in [0, 1]$. It is not known whether the results of this paper can be extended to the general case of $\lambda \in [0, 1]$. Prior research has suggested that larger values of $\lambda$ lead to superior results. In particular, an example of [1] and the approximation error bounds of [17, 18], both of which are restricted to the case of a fixed policy, suggest that approximation error is amplified by a factor of $1/(1 - \alpha)$ as $\lambda$ is changed from 1 to 0. The results of Sections 3 and 4 suggest that this factor vanishes if one considers a controlled process and performance loss rather than approximation error.

Whether the results of this paper can be extended to accommodate approximate value iteration with general linearly parameterized approximators remains an open issue. In this broader context, error and performance loss bounds of the kind offered by Theorem 2 are

unavailable, even when the invariant distribution is used to weight the projection. Such error and performance bounds *are* available, on the other hand, for the solution to a certain linear program [5, 6]. Whether a factor of $1/(1-\alpha)$ can similarly be eliminated from *these* bounds is an open issue.

Our results can be extended to accommodate an average cost objective, assuming that the MDP is communicating. With Boltzmann exploration, the equation of interest becomes

$$\Phi\tilde{r} = \Pi_{\pi_{\tilde{r}}^\epsilon}(T^\epsilon\Phi\tilde{r} - \tilde{\lambda}\mathbf{1}).$$

The variables include an estimate $\tilde{\lambda} \in \Re$ of the minimal average cost $\lambda^* \in \Re$ and an approximation $\Phi\tilde{r}$ of the optimal differential cost function $h^*$. The discount factor $\alpha$ is set to 1 in computing an $\epsilon$-greedy Boltzmann exploration policy as well as $T^\epsilon$. There is an average-cost version of temporal-difference learning for which any limit of convergence $(\tilde{\lambda}, \tilde{r})$ satisfies this equation [19, 20, 21]. Generalization of Theorem 2 does not lead to a useful result because the right-hand side of the bound becomes infinite as $\alpha$ approaches 1. On the other hand, generalization of Theorem 6 yields the first performance loss bound for approximate value iteration with an average-cost objective:

**Theorem 7** *For any communicating MDP with an average-cost objective, partition, and* $\epsilon > 0$, *if* $\Phi\tilde{r} = \Pi_{\pi_{\tilde{r}}^\epsilon}(T^\epsilon\Phi\tilde{r} - \tilde{\lambda}\mathbf{1})$ *then*

$$\lambda_{\mu_{\tilde{r}}^\epsilon} - \lambda^* \leq 2 \min_{r\in\Re^K} \|h^* - \Phi r\|_\infty + \epsilon.$$

Here, $\lambda_{\mu_{\tilde{r}}^\epsilon} \in \Re$ denotes the average cost under policy $\mu_{\tilde{r}}^\epsilon$, which is well-defined because the process is irreducible under an $\epsilon$-greedy Boltzmann exploration policy. This theorem can be proved by taking limits on the left and right-hand sides of the bound of Theorem 6. It is easy to see that the limit of the left-hand side is $\lambda_{\mu_{\tilde{r}}^\epsilon} - \lambda^*$. The limit of $\min_{r\in\Re^K} \|J^* - \Phi r\|_\infty$ on the right-hand side is $\min_{r\in\Re^K} \|h^* - \Phi r\|_\infty$. (This follows from the analysis of [3].)

### Acknowledgments

This material is based upon work supported by the National Science Foundation under Grant ECS-9985229 and by the Office of Naval Research under Grant MURI N00014-00-1-0637. The author's understanding of the topic benefited from collaborations with Dimitri Bertsekas, Daniela de Farias, and John Tsitsiklis. A full length version of this paper has been submitted to *Mathematics of Operations Research* and has benefited from a number of useful comments and suggestions made by reviewers.

## Footnotes

[1]By an *invariant state distribution* of a transition matrix $P$, we mean any probability distribution $\pi$ such that $\pi^T P = \pi^T$. In the event that $P_{\mu_{\tilde{r}}}$ has multiple invariant distributions, $\pi_{\tilde{r}}$ denotes an arbitrary choice.

### References

[1] D. P. Bertsekas. A counterexample to temporal-difference learning. *Neural Computation*, 7:270–279, 1994.

[2] D. P. Bertsekas and J. N. Tsitsiklis. *Neuro-Dynamic Programming*. Athena Scientific, Belmont, MA, 1996.

[3] D. Blackwell. Discrete dynamic programming. *Annals of Mathematical Statistics*, 33:719–726, 1962.

[4] D. P. de Farias and B. Van Roy. On the existence of fixed points for approximate value iteration and temporal-difference learning. *Journal of Optimization Theory and Applications*, 105(3), 2000.

[5] D. P. de Farias and B. Van Roy. Approximate dynamic programming via linear programming. In *Advances in Neural Information Processing Systems 14*. MIT Press, 2002.

[6] D. P. de Farias and B. Van Roy. The linear programming approach to approximate dynamic programming. *Operations Research*, 51(6):850–865, 2003.

[7] G. J. Gordon. Stable function approximation in dynamic programming. Technical Report CMU-CS-95-103, Carnegie Mellon University, 1995.

[8] G. J. Gordon. Stable function approximation in dynamic programming. In *Machine Learning: Proceedings of the Twelfth International Conference (ICML)*, San Francisco, CA, 1995.

[9] T. Jaakkola, M. I. Jordan, and S. P. Singh. On the Convergence of Stochastic Iterative Dynamic Programming Algorithms. *Neural Computation*, 6:1185–1201, 1994.

[10] S. P. Singh and R. C. Yee. An upper-bound on the loss from approximate optimal-value functions. *Machine Learning*, 1994.

[11] R. S. Sutton. *Temporal Credit Assignment in Reinforcement Learning*. PhD thesis, University of Massachusetts, Amherst, Amherst, MA, 1984.

[12] R. S. Sutton. Learning to predict by the methods of temporal differences. *Machine Learning*, 3:9–44, 1988.

[13] R. S. Sutton. On the virtues of linear learning and trajectory distributions. In *Proceedings of the Workshop on Value Function Approximation, Machine Learning Conference*, 1995.

[14] R. S. Sutton. Generalization in reinforcement learning: Successful examples using sparse coarse coding. In *Advances in Neural Information Processing Systems 8*, Cambridge, MA, 1996. MIT Press.

[15] J. N. Tsitsiklis. Asynchronous stochastic approximation and Q-learning. *Machine Learning*, 16:185–202, 1994.

[16] J. N. Tsitsiklis and B. Van Roy. Feature–based methods for large scale dynamic programming. *Machine Learning*, 22:59–94, 1996.

[17] J. N. Tsitsiklis and B. Van Roy. An analysis of temporal–difference learning with function approximation. *IEEE Transactions on Automatic Control*, 42(5):674–690, 1997.

[18] J. N. Tsitsiklis and B. Van Roy. Analysis of temporal-difference learning with function approximation. In *Advances in Neural Information Processing Systems 9*, Cambridge, MA, 1997. MIT Press.

[19] J. N. Tsitsiklis and B. Van Roy. Average cost temporal-difference learning. In *Proceedings of the IEEE Conference on Decision and Control*, 1997.

[20] J. N. Tsitsiklis and B. Van Roy. Average cost temporal-difference learning. *Automatica*, 35(11):1799–1808, 1999.

[21] J. N. Tsitsiklis and B. Van Roy. On average versus discounted reward temporal-difference learning. *Machine Learning*, 49(2-3):179–191, 2002.

[22] B. Van Roy. Performance loss bounds for approximate value iteration with state aggregation. Under review with *Mathematics of Operations Research*, available at www.stanford.edu/ bvr/psfiles/aggregation.pdf, 2005.

[23] C. J. C. H. Watkins. *Learning From Delayed Rewards*. PhD thesis, Cambridge University, Cambridge, UK, 1989.

[24] C. J. C. H. Watkins and P. Dayan. Q-learning. *Machine Learning*, 8:279–292, 1992.
